# Fast Learning from Non-i.i.d. Observations

**Ingo Steinwart**
Information Sciences Group CCS-3
Los Alamos National Laboratory
Los Alamos, NM 87545, USA
ingo@lanl.gov

**Andreas Christmann**
University of Bayreuth
Department of Mathematics
D-95440 Bayreuth
Andreas.Christmann@uni-bayreuth.de

## Abstract

We prove an oracle inequality for generic regularized empirical risk minimization algorithms learning from $\alpha$-mixing processes. To illustrate this oracle inequality, we use it to derive learning rates for some learning methods including least squares SVMs. Since the proof of the oracle inequality uses recent localization ideas developed for independent and identically distributed (i.i.d.) processes, it turns out that these learning rates are close to the optimal rates known in the i.i.d. case.

## 1 Introduction

In the past, most articles investigating statistical properties of learning algorithms assumed that the observed data was generated in an i.i.d. fashion. However, in many applications this assumption cannot be strictly justified since the sample points are intrinsically temporal and thus often weakly dependent. Typical examples for this phenomenon are applications where observations come from (suitably pre-processed) time series, i.e., for example, financial predictions, signal processing, system observation and diagnosis, and speech or text recognition. A set of natural and widely accepted notions for describing such weak dependencies[1] are mixing concepts such as $\alpha$-, $\beta$-, and $\phi$-mixing, since *a)* they offer a generalization to i.i.d. processes that is satisfied by various types of stochastic processes including Markov chains and many time series models, and *b)* they quantify the dependence in a conceptually simple way that is accessible to various types of analysis.

Because of these features, the machine learning community is currently in the process of appreciating and accepting these notions as the increasing number of articles in this direction shows. Probably the first work in this direction goes back to Yu [20], whose techniques for $\beta$-mixing processes inspired subsequent work such as [18, 10, 11], while the analysis of specific learning algorithms probably started with [9, 5, 8]. More recently, [7] established consistency of regularized boosting algorithms learning from $\beta$-mixing processes, while [15] established consistency of support vector machines (SVMs) learning from $\alpha$-mixing processes, which constitute the largest class of mixing processes. For the latter, [21] established generalization bounds for empirical risk minimization (ERM) and [19, 17] analyzed least squares support vector machines (LS-SVMs).

In this work, we establish a general oracle inequality for generic regularized learning algorithms and $\alpha$-mixing observations by combining a Bernstein inequality for such processes [9] with localization ideas for i.i.d. processes pioneered by [6] and refined by e.g. [1]. To illustrate this oracle inequality, we then use it to show learning rates for some algorithms including ERM over finite sets and LS-SVMs. In the ERM case our results match those in the i.i.d. case if one replaces the number of observations with the "effective number of observations", while, for LS-SVMs, our rates are at least quite close to the recently obtained optimal rates [16] for i.i.d. observations. However, the latter difference is not surprising, when considering the fact that [16] used heavy machinery from

empirical process theory such as Talagrand's inequality and localized Rademacher averages, while our results only use a light-weight argument based on Bernstein's inequality.

## 2 Definitions, Results, and Examples

Let $X$ be a measurable space and $Y \subset \mathbb{R}$ be closed. Furthermore, let $(\Omega, \mathcal{A}, \mu)$ be a probability space and $\mathcal{Z} := (Z_i)_{i \geq 1}$ be a stochastic process such that $Z_i : \Omega \to X \times Y$ for all $i \geq 1$. For $n \geq 1$, we further write $D_n := ((X_1, Y_1), \ldots, (X_n, Y_n)) := (Z_1, \ldots, Z_n)$ for a *training set* of length $n$ that is distributed according to the first $n$ components of $\mathcal{Z}$. Throughout this work, we assume that $\mathcal{Z}$ is *stationary*, i.e., the $(X \times Y)^n$-valued random variables $(Z_{i_1}, \ldots, Z_{i_n})$ and $(Z_{i_1+i}, \ldots, Z_{i_n+i})$ have the same distribution for all $n, i, i_1, \ldots, i_n \geq 1$. We further write $P$ for the distribution of one (and thus all) $Z_i$, i.e., for all measurable $A \subset X \times Y$, we have

$$P(A) = \mu\big(\{\omega \in \Omega : Z_i(\omega) \in A\}\big). \tag{1}$$

To learn from stationary processes whose components are not independent, [15] suggests that it is necessary to replace the independence assumption by a notion that still guarantees certain concentration inequalities. We will focus on $\alpha$-mixing, which is based on the *$\alpha$-mixing coefficients*

$$\alpha(\mathcal{Z}, \mu, n) := \sup\Big\{ \big|\mu(A \cap B) - \mu(A)\mu(B)\big| : i \geq 1,\ A \in \mathcal{A}_1^i \text{ and } B \in \mathcal{A}_{i+n}^\infty \Big\}, \qquad n \geq 1,$$

where $\mathcal{A}_1^i$ and $\mathcal{A}_{i+n}^\infty$ are the $\sigma$-algebras generated by $(Z_1, \ldots, Z_i)$ and $(Z_{i+n}, Z_{i+n+1}, \ldots)$, respectively. Throughout this work, we assume that the process $\mathcal{Z}$ is *geometrically $\alpha$-mixing*, that is

$$\alpha(\mathcal{Z}, \mu, n) \leq c \exp(-bn^\gamma), \qquad n \geq 1, \tag{2}$$

for some constants $b > 0$, $c \geq 0$, and $\gamma > 0$. Of course, i.i.d. processes satisfy (2) for $c = 0$ and all $b, \gamma > 0$. Moreover, several time series models such as ARMA and GARCH, which are often used to describe, e.g. financial data, satisfy (2) under natural conditions [4, Chapter 2.6.1], and the same is true for many Markov chains including some dynamical systems perturbed by dynamic noise, see e.g. [18, Chapter 3.5]. An extensive and thorough account on mixing concepts including stronger mixing notions such as $\beta$- and $\phi$-mixing is provided by [3].

Let us now describe the learning algorithms we are interested in. To this end, we assume that we have a *hypothesis set* $\mathcal{F}$ consisting of *bounded* measurable functions $f : X \to \mathbb{R}$ that is pre-compact with respect to the supremum norm $\| \cdot \|_\infty$, i.e., for all $\varepsilon > 0$, the covering numbers

$$\mathcal{N}(\mathcal{F}, \| \cdot \|_\infty, \varepsilon) := \inf\Big\{ n \geq 1 : \exists f_1, \ldots, f_n \in \mathcal{F} \text{ such that } \mathcal{F} \subset \bigcup_{i=1}^n B(f_i, \varepsilon) \Big\}$$

are finite, where $B(f_i, \varepsilon) := \{f \in \ell_\infty(X) : \|f - f_i\|_\infty \leq \varepsilon\}$ denotes the $\varepsilon$-ball with center $f_i$ in the space $\ell_\infty(X)$ of bounded functions $f : X \to \mathbb{R}$. Moreover, we assume that we have a *regularizer*, that is, a function $\Upsilon : \mathcal{F} \to [0, \infty)$. Following [13, Definition 2.22], we further say that a function $L : X \times Y \times \mathbb{R} \to [0, \infty)$ is a *loss* that can be *clipped* at some $M > 0$, if $L$ is measurable and

$$L(x, y, \bar{t}) \leq L(x, y, t), \qquad (x, y, t) \in X \times Y \times \mathbb{R}, \tag{3}$$

where $\bar{t}$ denotes the clipped value of $t$ at $\pm M$, that is $\bar{t} := t$ if $t \in [-M, M]$, $\bar{t} := -M$ if $t < -M$, and $\bar{t} := M$ if $t > M$. Various often used loss functions can be clipped. For example, if $Y := \{-1, 1\}$ and $L$ is a convex, margin-based loss represented by $\varphi : \mathbb{R} \to [0, \infty)$, that is $L(y, t) = \varphi(yt)$ for all $y \in Y$ and $t \in \mathbb{R}$, then $L$ can be clipped, if and only if $\varphi$ has a global minimum, see [13, Lemma 2.23]. In particular, the hinge loss, the least squares loss for classification, and the squared hinge loss can be clipped, but the logistic loss for classification and the AdaBoost loss cannot be clipped. On the other hand, [12] established a simple technique, which is similar to inserting a small amount of noise into the labeling process, to construct a clippable modification of an arbitrary convex, margin-based loss. Moreover, if $Y := [-M, M]$ and $L$ is a convex, distance-based loss represented by some $\psi : \mathbb{R} \to [0, \infty)$, that is $L(y, t) = \psi(y - t)$ for all $y \in Y$ and $t \in \mathbb{R}$, then $L$ can be clipped whenever $\psi(0) = 0$, see again [13, Lemma 2.23]. In particular, the least squares loss and the pinball loss used for quantile regression can be clipped, if the space of labels $Y$ is bounded.

Given a loss function $L$ and an $f : X \to \mathbb{R}$, we often use the notation $L \circ f$ for the function $(x, y) \mapsto L(x, y, f(x))$. Moreover, the $L$-risk is defined by

$$\mathcal{R}_{L,P}(f) := \int_{X \times Y} L(x, y, f(x))\, dP(x, y),$$

and the minimal $L$-risk is $\mathcal{R}^*_{L,P} := \inf\{\mathcal{R}_{L,P}(f) \,|\, f : X \to \mathbb{R}\}$. In addition, a function $f^*_{L,P}$ satisfying $\mathcal{R}_{L,P}(f^*_{L,P}) = \mathcal{R}^*_{L,P}$ is called a Bayes decision function. Finally, we denote *empirical risks* based on $D_n$ by $\mathcal{R}_{L,D_n}(f)$, that is, for a realization of $D_n(\omega)$ of the training set $D_n$ we have

$$\mathcal{R}_{L,D_n(\omega)}(f) = \frac{1}{n}\sum_{i=1}^{n} L\big(X_i(\omega), Y_i(\omega), f(X_i(\omega))\big).$$

Given a regularizer $\Upsilon : \mathcal{F} \to [0, \infty)$, a clippable loss, and an accuracy $\delta \geq 0$, we consider learning methods that, for all $n \geq 1$, produce a decision function $f_{D_n, \Upsilon} \in \mathcal{F}$ satisfying

$$\Upsilon(f_{D_n,\Upsilon}) + \mathcal{R}_{L,D_n}(\bar{f}_{D_n,\Upsilon}) \leq \inf_{f \in \mathcal{F}}\Big(\Upsilon(f) + \mathcal{R}_{L,D_n}(f)\Big) + \delta. \tag{4}$$

Note that methods such as SVMs (see below) that minimize the right-hand side of (4) *exactly*, satisfy (4), because of (3). The following theorem, which is our main result, establishes an oracle inequality for methods (4), when the training data is generated by $\mathcal{Z}$.

**Theorem 2.1** *Let $L : X \times Y \times \mathbb{R} \to [0, \infty)$ be a loss that can be clipped at $M > 0$ and that satisfies $L(x, y, 0) \leq 1$, $L(x, y, t) \leq B$, and*

$$\big|L(x, y, t) - L(x, y, t')\big| \leq |t - t'| \tag{5}$$

*for all $(x, y) \in X \times Y$ and $t, t' \in [-M, M]$, where $B > 0$ is some constant. Moreover, let $\mathcal{Z} := (Z_i)_{i \geq 1}$ be an $X \times Y$-valued process that satisfies (2), and $P$ be defined by (1). Assume that there exist a Bayes decision function $f^*_{L,P}$ and constants $\vartheta \in [0, 1]$ and $V \geq B^{2-\vartheta}$ such that*

$$\mathbb{E}_P\big(L \circ \bar{f} - L \circ f^*_{L,P}\big)^2 \leq V \cdot \big(\mathbb{E}_P(L \circ \bar{f} - L \circ f^*_{L,P})\big)^\vartheta, \qquad f \in \mathcal{F}, \tag{6}$$

*where $\mathcal{F}$ is a hypothesis set and $L \circ f$ denotes the function $(x, y) \mapsto L(x, y, f(x))$. Finally, let $\Upsilon : \mathcal{F} \to [0, \infty)$ be a regularizer, $f_0 \in \mathcal{F}$ be a fixed function and $B_0 \geq B$ be a constant such that $\|L \circ f_0\|_\infty \leq B_0$. Then, for all fixed $\varepsilon > 0$, $\delta \geq 0$, $\tau > 0$, and $n \geq \max\{b/8, 2^{2+5/\gamma}b^{-1/\gamma}\}$, every learning method defined by (4) satisfies with probability $\mu$ not less than $1 - 3Ce^{-\tau}$:*

$$\Upsilon(f_{D_n,\Upsilon}) + \mathcal{R}_{L,P}(\bar{f}_{D_n,\Upsilon}) - \mathcal{R}^*_{L,P} \;<\; 3\big(\Upsilon(f_0) + \mathcal{R}_{L,P}(f_0) - \mathcal{R}^*_{L,P}\big) + \frac{4B_0 c_B \tau}{n^\alpha} + 4\varepsilon + 2\delta$$

$$+ \left(\frac{36 c_\sigma V(\tau + \ln \mathcal{N}(\mathcal{F}, \|\cdot\|_\infty, \varepsilon))}{n^\alpha}\right)^{1/(2-\vartheta)},$$

*where $\alpha := \frac{\gamma}{\gamma+1}$, $C := 1 + 4e^{-2}c$, $c_\sigma := (\frac{8^{2+\gamma}}{b})^{1/(1+\gamma)}$, and $c_B := c_\sigma/3$.*

Before we illustrate this theorem by a few examples, let us briefly discuss the variance bound (6). For example, if $Y = [-M, M]$ and $L$ is the least squares loss, then it is well-known that (6) is satisfied for $V := 16M^2$ and $\vartheta = 1$, see e.g. [13, Example 7.3]. Moreover, under some assumptions on the distribution $P$, [14] established a variance bound of the form (6) for the so-called pinball loss used for quantile regression. In addition, for the hinge loss, (6) is satisfied for $\vartheta := q/(q+1)$, if Tsybakov's noise assumption holds for $q$, see [13, Theorem 8.24]. Finally, based on [2], [12] established a variance bound with $\vartheta = 1$ for the earlier mentioned clippable modifications of strictly convex, twice continuously differentiable margin-based loss functions.

One might wonder, why the constant $B_0$ is necessary in Theorem 2.1, since appearently it only adds further complexity. However, a closer look reveals that the constant $B$ only bounds functions of the form $L \circ \bar{f}$, while $B_0$ bounds the function $L \circ f_0$ for an *unclipped* $f_0 \in \mathcal{F}$. Since we do not assume that all $f \in \mathcal{F}$ satisfy $\bar{f} = f$, we conclude that in general $B_0$ is necessary. We refer to Examples 2.4 and 2.5 for situations, where $B_0$ is significantly larger than $B$.

Let us now consider a few examples of learning methods to which Theorem 2.1 applies. The first one is empirical risk minimization over a finite set.

**Example 2.2** Let the hypothesis set $\mathcal{F}$ be finite and $\Upsilon(f) = 0$ for all $f \in \mathcal{F}$. Moreover, assume that $\|f\|_\infty \leq M$ for all $f \in \mathcal{F}$. Then, for accuracy $\delta := 0$, the learning method described by (4) is ERM, and Theorem 2.1 provides, by some simple estimates, the oracle inequality

$$\mathcal{R}_{L,P}(f_{D_n,\Upsilon}) - \mathcal{R}^*_{L,P} < 3 \inf_{f \in \mathcal{F}}\big(\mathcal{R}_{L,P}(f) - \mathcal{R}^*_{L,P}\big) + \left(\frac{36 c_\sigma V(\tau + \ln|\mathcal{F}|)}{n^\alpha}\right)^{1/(2-\vartheta)} + \frac{4B c_B \tau}{n^\alpha}.$$

Besides constants, this oracle inequality is an exact analogue to the standard oracle inequality for ERM learning from i.i.d. processes, [13, Theorem 7.2].                                                    ◁

Before we present another example, let us first reformulate Theorem 2.1 for the case that the involved covering numbers have a certain polynomial behavior.

**Corollary 2.3** *Consider the situation of Theorem 2.1 and additionally assume that there exist constants $a > 0$ and $p \in (0, 1]$ such that*

$$\ln \mathcal{N}(\mathcal{F}, \| \cdot \|_\infty, \varepsilon) \leq a\, \varepsilon^{-2p}\,, \qquad\qquad \varepsilon > 0\,.$$

*Then there is $c_{p,\vartheta} > 0$ only depending on $p$ and $\vartheta$ such that the inequality of Theorem 2.1 reduces to*

$$\Upsilon(f_{D_n},\Upsilon) + \mathcal{R}_{L,P}(\bar{f}_{D_n},\Upsilon) - \mathcal{R}^*_{L,P} < 3\big(\Upsilon(f_0) + \mathcal{R}_{L,P}(f_0) - \mathcal{R}^*_{L,P}\big) + c_{p,\vartheta}\left(\frac{c_\sigma V a}{n^\alpha}\right)^{1/(2+2p-\vartheta)}$$

$$+ \left(\frac{36 c_\sigma V \tau}{n^\alpha}\right)^{1/(2-\vartheta)} + \frac{4B_0 c_B \tau}{n^\alpha} + 2\delta\,.$$

For the learning rates considered in the following examples, the exact value of $c_{p,\vartheta}$ is of no importance. However, a careful numerical analysis shows that $c_{p,\vartheta} \leq 40$ for all $p \in (0, 1]$ and $\vartheta \in [0, 1]$.

Corollary 2.3 can be applied to various methods including e.g. SVMs with the hinge loss or the pinball loss, and regularized boosting algorithms. For the latter, we refer to e.g. [2] for some learning rates in the i.i.d. case and to [7] for a consistency result in the case of geometrically $\beta$-mixing observations. Unfortunately, a detailed exposition of the learning rates resulting from Corollary 2.3 for all these algorithms is clearly out of scope this paper, and hence we will only discuss learning rates for LS-SVMs. However, the *only* reason we picked LS-SVMs is that they are one of the few methods for which both rates for learning from $\alpha$-mixing processes and optimal rates in the i.i.d. case are known. By considering LS-SVMs we can thus assess the sharpness of our results. Let us begin by briefly recalling LS-SVMs. To this end, let $X$ be a compact metric space and $k$ be a continuous kernel on $X$ with reproducing kernel Hilbert space (RKHS) $H$. Given a regularization parameter $\lambda > 0$ and the least squares loss $L(y, t) := (y - t)^2$, the LS-SVM finds the unique solution

$$f_{D_n,\lambda} = \arg \min_{f \in H} \big(\lambda \|f\|_H^2 + \mathcal{R}_{L,D_n}(f)\big)\,.$$

To describe the approximation properties of $H$, we further need the approximation error function

$$A(\lambda) := \inf_{f \in H} \big(\lambda \|f\|_H^2 + \mathcal{R}_{L,P}(f) - \mathcal{R}^*_{L,P}\big)\,, \qquad\qquad \lambda > 0\,.$$

**Example 2.4 (Rates for least squares SVMs)** Let $X$ be a compact metric space, $Y = [-1, 1]$, and $\mathcal{Z}$ and $P$ as above. Furthermore, let $L$ be the least squares loss and $H$ be the RKHS of a continuous kernel $k$ over $X$. Assume that the closed unit ball $B_H$ of $H$ satisfies

$$\ln \mathcal{N}(B_H, \| \cdot \|_\infty, \varepsilon) \leq a\, \varepsilon^{-2p}\,, \qquad\qquad \varepsilon > 0, \qquad\qquad (7)$$

where $a > 0$ and $p \in (0, 1]$ are some constants. In addition, assume that the approximation error function satisfies $A(\lambda) \leq c\lambda^\beta$ for some $c > 0$, $\beta \in (0, 1]$, and all $\lambda > 0$. We define

$$\rho := \min\Big\{\beta, \frac{\beta}{\beta + 2p\beta + p}\Big\}\,.$$

Then Corollary 2.3 applied to $\mathcal{F} := \lambda^{-1/2} B_H$ shows that the LS-SVM using $\lambda_n := n^{-\alpha\rho/\beta}$ learns with rate $n^{-\alpha\rho}$. Let us compare this rate with other recent results: [17] establishes the learning rate

$$n^{-\frac{2\beta}{\beta+3}}\,,$$

whenever (2) is satisfied for some $\alpha$. At first glance, this rate looks stronger, since it is independent of $\alpha$. However, a closer look shows that it depends on the confidence level $1 - 3Ce^{-\tau}$ by a factor of $e^\tau$ rather than by the factor of $\tau$ appearing in our analysis, and hence these rates are not comparable. Moreover, in the case $\alpha = 1$, our rates are still faster whenever $p \in (0, 1/3]$, which is e.g. satisfied

for sufficiently smooth kernels, see e.g. [13, Theorem 6.26]. Moreover, [19] has recently established the rate

$$n^{-\frac{\alpha\beta}{2p+1}},\tag{8}$$

which is faster than ours, if and only if $\beta > \frac{1+p}{1+2p}$. In particular, for highly smooth kernels such as the Gaussian RBF kernels, where $p$ can be chosen arbitrarily close to 0, their rate is never faster. Moreover, [19] requires knowing $\alpha$, which, as we will briefly discuss in Remark 2.6, is not the case for our rates. In this regard, it is interesting to note that their iterative proof procedure, see [13, Chapter 7.1] for a generic description of this technique, can also be applied to our oracle inequality. The resulting rate is essentially $n^{-\alpha\min\{\beta,\beta/(\beta+p\beta+p)\}}$, which is *always* faster than (8). Due to space constraints and the fact that these rates require knowing $\alpha$ and $\beta$, we skip a detailed exposition. Finally, both [19] and [17] only consider LS-SVMs, while Theorem 2.1 applies to various learning methods. ◁

**Example 2.5 (Almost optimal rates for least squares SVMs)** Consider the situation of Example 2.4, and additionally assume that there exists a constant $C_p > 0$ such that

$$\|f\|_\infty \le C_p\, \|f\|_H^p\|f\|_{L_2(P_X)}^{1-p}, \qquad\qquad f \in H.\tag{9}$$

As in [16], we can then bound $B_0 \le \lambda^{(\beta-1)p}$, and hence the SVM using $\lambda_n := n^{-\frac{\alpha}{\beta+2p\beta+p}}$ learns with rate

$$n^{-\frac{\alpha\beta}{\beta+2p\beta+p}},$$

compared to the optimal rate $n^{-\frac{\beta}{\beta+p}}$ in the i.i.d. case, see [16]. In particular, if $H = W^m(X)$ is a Sobolev space over $X \subset \mathbb{R}^d$ with smoothness $m > d/2$, and the marginal distribution $P_X$ is absolutely continuous with respect to the uniform distribution, where corresponding density is bounded away from 0 and $\infty$, then (7) and (9) are satisfied for $p := \frac{d}{2m}$. Moreover, the assumption on the approximation error function is satisfied for $\beta := s/m$, whenever $f_{L,P}^* \in W^s(X)$ and $s \in (d/2, m]$. Consequently, the resulting learning rate is

$$n^{-\frac{2s\alpha}{2s+d+2ds/m}},$$

which in the i.i.d. case, where $\alpha = 1$, is worse than the optimal rate $n^{-\frac{2s}{2s+d}}$ by the term $2ds/m$. Note that this difference can be made arbitrarily small by picking a sufficiently large $m$. Unfortunately, we do not know, whether the extra term $2ds/m$ is an artifact of our proof techniques, which are relatively light-weighted compared to the heavy machinery used in the i.i.d. case. Similarly, we do not know, whether the used Bernstein inequality for $\alpha$-mixing processes, see Theorem 3.1, is optimal, but it is the best inequality we could find in the literature. However, if there is, or will be, a better version of this inequality, our oracle inequalities can be easily improved, since our techniques only require a generic form of Bernstein's inequality. ◁

**Remark 2.6** In the examples above, the rates were achieved by picking particular regularization sequences that depend on both $\alpha$ and $\beta$, which in turn, are almost never known in practice. Fortunately, there exists an easy way to achieve the above rates without such knowledge. Indeed, let us assume we pick a polynomially growing $n^{-1/p}$-net $\Lambda_n$ of $(0,1]$, split the training sample $D_n$ into two (almost) equally sized and consecutive parts $D_n^{(1)}$ and $D_n^{(2)}$, compute $f_{D_n^{(1)},\lambda}$ for all $\lambda \in \Lambda_n$, and pick a $\lambda^* \in \Lambda_n$ whose $f_{D_n^{(1)},\lambda^*}$ minimizes the $\mathcal{R}_{L,D_n^{(2)}}$-risk over $\Lambda_n$. Then combining Example 2.2 with the oracle inequality of Corollary 2.3 for LS-SVMs shows that the learning rates of the Examples 2.4 and 2.5 are also achieved by this training-validation approach. Although the proof is a straightforward modification of [13, Theorem 7.24], it is out of the page limit of this paper. ◁

## 3  Proofs

In the following, $\lfloor t \rfloor$ denotes the largest integer $n$ satisfying $n \le t$, and similarly, $\lceil t \rceil$ denotes the smallest integer $n$ satisfying $n \ge t$.

The key result we need to prove the oracle inequality of Theorem 2.1 is the following Bernstein type inequality for geometrically $\alpha$-mixing processes, which was established in [9, Theorem 4.3]:

**Theorem 3.1** *Let $\mathcal{Z} := (Z_i)_{i \geq 1}$ be an $X \times Y$-valued stochastic process that satisfies (2) and $P$ be defined by (1). Furthermore, let $h : X \times Y \to \mathbb{R}$ be a bounded measurable function for which there exist constants $B > 0$ and $\sigma \geq 0$ such that $\mathbb{E}_P h = 0$, $\mathbb{E}_P h^2 \leq \sigma^2$, and $\|h\|_\infty \leq B$. For $n \geq 1$ we define*

$$n^{(\gamma)} := \left\lfloor n \left\lceil \left( \frac{8n}{b} \right)^{\frac{1}{\gamma+1}} \right\rceil^{-1} \right\rfloor .$$

*Then, for all $n \geq 1$ and all $\varepsilon > 0$, we have*

$$\mu\left( \left\{ \omega \in \Omega : \frac{1}{n} \sum_{i=1}^n h(Z_i(\omega)) \geq \varepsilon \right\} \right) \leq \left( 1 + 4e^{-2}c \right) e^{-\frac{3\varepsilon^2 n^{(\gamma)}}{6\sigma^2 + 2\varepsilon B}} . \tag{10}$$

Before we prove Theorem 2.1, we need to slightly modify (10). To this end, we first observe that $\lceil t \rceil \leq 2t$ for all $t \geq 1$ and $\lfloor t \rfloor \geq t/2$ for all $t \geq 2$. From this it is easy to conclude that, for all $n$ satisfying $n \geq n_0 := \max\{b/8, 2^{2+5/\gamma} b^{-1/\gamma}\}$, we have

$$n^{(\gamma)} \geq 2^{-\frac{2\gamma+5}{\gamma+1}} b^{\frac{1}{\gamma+1}} n^\alpha ,$$

where $\alpha := \frac{\gamma}{\gamma+1}$. For $C := 1 + 4e^{-2}c$, $c_\sigma := (\frac{8^{2+\gamma}}{b})^{1/(1+\gamma)}$, and $c_B := c_\sigma/3$, we thus obtain

$$\mu\left( \left\{ \omega \in \Omega : \frac{1}{n} \sum_{i=1}^n h(Z_i(\omega)) \geq \varepsilon \right\} \right) \leq C e^{-\tau} , \qquad n \geq n_0,$$

where $\tau := \frac{\varepsilon^2 n^\alpha}{c_\sigma \sigma^2 + \varepsilon c_B B}$. Simple transformations and estimations then yield

$$\mu\left( \left\{ \omega \in \Omega : \frac{1}{n} \sum_{i=1}^n h(Z_i(\omega)) \geq \sqrt{\frac{\tau c_\sigma \sigma^2}{n^\alpha}} + \frac{c_B B \tau}{n^\alpha} \right\} \right) \leq C e^{-\tau} \tag{11}$$

for all $n \geq \max\{b/8, 2^{2+5/\gamma} b^{-1/\gamma}\}$ and $\tau > 0$. In the following, we will use only this inequality. In addition, we will need the following simple and well-known lemma:

**Lemma 3.2** *For $q \in (1, \infty)$, define $q' \in (1, \infty)$ by $1/q + 1/q' = 1$. Then, for all $a, b \geq 0$, we have $(qa)^{2/q}(q'b)^{2/q'} \leq (a+b)^2$ and $ab \leq a^q/q + b^{q'}/q'$.*

***Proof of Theorem 2.1:*** For $f : X \to \mathbb{R}$ we define $h_f := L \circ f - L \circ f_{L,P}^*$. By the definition of $f_{D_n, \Upsilon}$, we then have $\Upsilon(f_{D_n, \Upsilon}) + \mathbb{E}_{D_n} h_{\bar{f}_{D_n, \Upsilon}} \leq \Upsilon(f_0) + \mathbb{E}_{D_n} h_{f_0} + \delta$, and consequently we obtain

$$
\begin{aligned}
&\Upsilon(f_{D_n, \Upsilon}) + \mathcal{R}_{L,P}(\bar{f}_{D_n, \Upsilon}) - \mathcal{R}_{L,P}^* \\
&= \Upsilon(f_{D_n, \Upsilon}) + \mathbb{E}_P h_{\bar{f}_{D_n, \Upsilon}} \\
&\leq \Upsilon(f_0) + \mathbb{E}_{D_n} h_{f_0} - \mathbb{E}_{D_n} h_{\bar{f}_{D_n, \Upsilon}} + \mathbb{E}_P h_{\bar{f}_{D_n, \Upsilon}} + \delta \\
&= (\Upsilon(f_0) + \mathbb{E}_P h_{f_0}) + (\mathbb{E}_{D_n} h_{f_0} - \mathbb{E}_P h_{f_0}) + (\mathbb{E}_P h_{\bar{f}_{D_n, \Upsilon}} - \mathbb{E}_{D_n} h_{\bar{f}_{D_n, \Upsilon}}) + \delta .
\end{aligned} \tag{12}
$$

Let us first bound the term $\mathbb{E}_{D_n} h_{f_0} - \mathbb{E}_P h_{f_0}$. To this end, we further split this difference into

$$\mathbb{E}_{D_n} h_{f_0} - \mathbb{E}_P h_{f_0} = \left( \mathbb{E}_{D_n}(h_{f_0} - h_{\bar{f}_0}) - \mathbb{E}_P(h_{f_0} - h_{\bar{f}_0}) \right) + \left( \mathbb{E}_{D_n} h_{\bar{f}_0} - \mathbb{E}_P h_{\bar{f}_0} \right) . \tag{13}$$

Now $L \circ f_0 - L \circ \bar{f}_0 \geq 0$ implies $h_{f_0} - h_{\bar{f}_0} = L \circ f_0 - L \circ \bar{f}_0 \in [0, B_0]$, and hence we obtain

$$\mathbb{E}_P \left( (h_{f_0} - h_{\bar{f}_0}) - \mathbb{E}_P(h_{f_0} - h_{\bar{f}_0}) \right)^2 \leq \mathbb{E}_P(h_{f_0} - h_{\bar{f}_0})^2 \leq B_0 \mathbb{E}_P(h_{f_0} - h_{\bar{f}_0}) .$$

Inequality (11) applied to $h := (h_{f_0} - h_{\bar{f}_0}) - \mathbb{E}_P(h_{f_0} - h_{\bar{f}_0})$ thus shows that

$$\mathbb{E}_{D_n}(h_{f_0} - h_{\bar{f}_0}) - \mathbb{E}_P(h_{f_0} - h_{\bar{f}_0}) < \sqrt{\frac{\tau c_\sigma B_0 \mathbb{E}_P(h_{f_0} - h_{\bar{f}_0})}{n^\alpha}} + \frac{c_B B_0 \tau}{n^\alpha}$$

holds with probability $\mu$ not less than $1 - Ce^{-\tau}$. Moreover, using $\sqrt{ab} \leq \frac{a}{2} + \frac{b}{2}$, we find

$$\sqrt{n^{-\alpha} \tau c_\sigma B_0 \mathbb{E}_P(h_{f_0} - h_{\bar{f}_0})} \leq \mathbb{E}_P(h_{f_0} - h_{\bar{f}_0}) + n^{-\alpha} c_\sigma B_0 \tau/4 ,$$

and consequently we have with probability $\mu$ not less than $1 - Ce^{-\tau}$ that

$$\mathbb{E}_{D_n}(h_{f_0} - h_{\bar{f}_0}) - \mathbb{E}_P(h_{f_0} - h_{\bar{f}_0}) < \mathbb{E}_P(h_{f_0} - h_{\bar{f}_0}) + \frac{7c_B B_0 \tau}{4n^\alpha} \,. \tag{14}$$

In order to bound the remaining term in (13), that is $\mathbb{E}_{D_n} h_{\bar{f}_0} - \mathbb{E}_P h_{\bar{f}_0}$, we first observe that (5) implies $\|h_{\bar{f}_0}\|_\infty \leq B$, and hence we have $\|h_{\bar{f}_0} - \mathbb{E}_P h_{\bar{f}_0}\|_\infty \leq 2B$. Moreover, (6) yields

$$\mathbb{E}_P(h_{\bar{f}_0} - \mathbb{E}_P h_{\bar{f}_0})^2 \leq \mathbb{E}_P h_{\bar{f}_0}^2 \leq V(\mathbb{E}_P h_{\bar{f}_0})^\vartheta \,.$$

In addition, if $\vartheta \in (0,1]$, Lemma 3.2 implies for $q := \frac{2}{2-\vartheta}$, $q' := \frac{2}{\vartheta}$, $a := (n^{-\alpha} c_\sigma 2^{-\vartheta} \vartheta^\vartheta V \tau)^{1/2}$, and $b := (2\vartheta^{-1}\mathbb{E}_P h_{\bar{f}_0})^{\vartheta/2}$, that

$$\sqrt{\frac{c_\sigma V \tau (\mathbb{E}_P h_{\bar{f}_0})^\vartheta}{n^\alpha}} \leq \left(1 - \frac{\vartheta}{2}\right)\left(\frac{c_\sigma 2^{-\vartheta} \vartheta^\vartheta V \tau}{n^\alpha}\right)^{\frac{1}{2-\vartheta}} + \mathbb{E}_P h_{\bar{f}_0} \leq \left(\frac{c_\sigma V \tau}{n^\alpha}\right)^{\frac{1}{2-\vartheta}} + \mathbb{E}_P h_{\bar{f}_0} \,.$$

Since $\mathbb{E}_P h_{\bar{f}_0} \geq 0$, this inequality also holds for $\vartheta = 0$, and hence (11) shows that we have

$$\mathbb{E}_{D_n} h_{\bar{f}_0} - \mathbb{E}_P h_{\bar{f}_0} < \mathbb{E}_P h_{\bar{f}_0} + \left(\frac{c_\sigma V \tau}{n^\alpha}\right)^{\frac{1}{2-\vartheta}} + \frac{2c_B B \tau}{n^\alpha} \tag{15}$$

with probability $\mu$ not less than $1 - Ce^{-\tau}$. By combining this estimate with (14) and (13), we now obtain that with probability $\mu$ not less than $1 - 2Ce^{-\tau}$ we have

$$\mathbb{E}_{D_n} h_{\bar{f}_0} - \mathbb{E}_P h_{\bar{f}_0} < \mathbb{E}_P h_{\bar{f}_0} + \left(\frac{c_\sigma V \tau}{n^\alpha}\right)^{\frac{1}{2-\vartheta}} + \frac{2c_B B \tau}{n^\alpha} + \frac{7c_B B_0 \tau}{4n^\alpha} \,, \tag{16}$$

i.e., we have established a bound on the second term in (12).

Let us now fix a minimal $\varepsilon$-net $\mathcal{C}$ of $\mathcal{F}$, that is, an $\varepsilon$-net of cardinality $|\mathcal{C}| = \mathcal{N}(\mathcal{F}, \|\cdot\|_\infty, \varepsilon)$. Let us first consider the case $n^\alpha < 3c_B(\tau + \ln|\mathcal{C}|)$. Combining (16) with (12) and using $B \leq B_0$, $B^{2-\vartheta} \leq V$, $3c_B \leq c_\sigma$, $2 \leq 4^{1/(2-\vartheta)}$, and $\mathbb{E}_P h_{\bar{f}_{D_n},\Upsilon} - \mathbb{E}_{D_n} h_{\bar{f}_{D_n},\Upsilon} \leq 2B$, we then find

$$\Upsilon(f_{D_n},\Upsilon) + \mathcal{R}_{L,P}(f_{D_n},\Upsilon) - \mathcal{R}_{L,P}^*$$

$$\leq \Upsilon(f_0) + 2\mathbb{E}_P h_{f_0} + \left(\frac{c_\sigma V \tau}{n^\alpha}\right)^{\frac{1}{2-\vartheta}} + \frac{2c_B B \tau}{n^\alpha} + \frac{7c_B B_0 \tau}{4n^\alpha} + (\mathbb{E}_P h_{\bar{f}_{D_n},\Upsilon} - \mathbb{E}_{D_n} h_{\bar{f}_{D_n},\Upsilon}) + \delta$$

$$\leq \Upsilon(f_0) + 2\mathbb{E}_P h_{f_0} + \left(\frac{c_\sigma V(\tau + \ln|\mathcal{C}|)}{n^\alpha}\right)^{\frac{1}{2-\vartheta}} + \frac{4c_B B_0 \tau}{n^\alpha} + 2B\left(\frac{c_\sigma(\tau + \ln|\mathcal{C}|)}{n^\alpha}\right)^{\frac{1}{2-\vartheta}} + \delta$$

$$\leq 3\Upsilon(f_0) + 3\mathbb{E}_P h_{f_0} + \left(\frac{36c_\sigma V(\tau + \ln|\mathcal{C}|)}{n^\alpha}\right)^{\frac{1}{2-\vartheta}} + \frac{4c_B B_0 \tau}{n^\alpha} + \delta$$

with probability $\mu$ not less than $1 - 2e^{-\tau}$. It thus remains to consider the case $n^\alpha \geq 3c_B(\tau + \ln|\mathcal{C}|)$. To establish a non-trivial bound on the term $\mathbb{E}_P h_{\bar{f}_D} - \mathbb{E}_{D_n} h_{\bar{f}_D}$ in (12), we define functions

$$g_{f,r} := \frac{\mathbb{E}_P h_{\bar{f}} - h_{\bar{f}}}{\mathbb{E}_P h_{\bar{f}} + r} \,, \qquad f \in \mathcal{F} \,,$$

where $r > 0$ is a real number to be fixed later. For $f \in \mathcal{F}$, we then have $\|g_{f,r}\|_\infty \leq 2Br^{-1}$, and for $\vartheta > 0$, $q := \frac{2}{2-\vartheta}$, $q' := \frac{2}{\vartheta}$, $a := r$, and $b := \mathbb{E}_P h_{\bar{f}} \neq 0$, the first inequality of Lemma 3.2 yields

$$\mathbb{E}_P g_{f,r}^2 \leq \frac{\mathbb{E}_P h_{\bar{f}}^2}{(\mathbb{E}_P h_{\bar{f}} + r)^2} \leq \frac{(2-\vartheta)^{2-\vartheta} \vartheta^\vartheta \, \mathbb{E}_P h_{\bar{f}}^2}{4r^{2-\vartheta}(\mathbb{E}_P h_{\bar{f}})^\vartheta} \leq V r^{\vartheta-2} \,. \tag{17}$$

Moreover, for $\vartheta \in (0,1]$ and $\mathbb{E}_P h_{\bar{f}} = 0$, we have $\mathbb{E}_P h_{\bar{f}}^2 = 0$ by the variance bound (6), which in turn implies $\mathbb{E}_P g_{f,r}^2 \leq V r^{\vartheta-2}$. Finally, it is not hard to see that $\mathbb{E}_P g_{f,r}^2 \leq V r^{\vartheta-2}$ also holds for $\vartheta = 0$. Now, (11) together with a simple union bound yields

$$\mu\left(D_n \in (X \times Y)^n : \sup_{f \in \mathcal{C}} \mathbb{E}_{D_n} g_{f,r} < \sqrt{\frac{c_\sigma V \tau}{n^\alpha r^{2-\vartheta}}} + \frac{2c_B B \tau}{n^\alpha r}\right) \geq 1 - C|\mathcal{C}|e^{-\tau} \,,$$

and consequently we see that, with probability $\mu$ not less than $1 - C\,|\mathcal{C}|\,e^{-\tau}$, we have

$$\mathbb{E}_P h_{\bar{f}} - \mathbb{E}_{D_n} h_{\bar{f}} < \left(\mathbb{E}_P h_{\bar{f}} + r\right)\left(\sqrt{\frac{c_\sigma V \tau}{n^\alpha r^{2-\vartheta}}} + \frac{2c_B B \tau}{n^\alpha r}\right) \tag{18}$$

for all $f \in \mathcal{C}$. Since $f_{D_n,\Upsilon} \in \mathcal{F}$, there now exists an $f_{D_n} \in \mathcal{C}$ with $\|f_{D_n,\Upsilon} - f_{D_n}\|_\infty \leq \varepsilon$. By the assumed Lipschitz continuity of $L$ the latter implies

$$\left|h_{\bar{f}_{D_n}}(x,y) - h_{\bar{f}_{D_n},\Upsilon}(x,y)\right| \leq \left|\bar{f}_{D_n}(x) - \bar{f}_{D_n,\Upsilon}(x)\right| \leq \left|f_{D_n}(x) - f_{D_n,\Upsilon}(x)\right| \leq \varepsilon$$

for all $(x,y) \in X \times Y$. Combining this with (18), we obtain

$$\mathbb{E}_P h_{\bar{f}_{D_n},\Upsilon} - \mathbb{E}_{D_n} h_{\bar{f}_{D_n},\Upsilon} < \left(\mathbb{E}_P h_{\bar{f}} + \varepsilon + r\right)\left(\sqrt{\frac{c_\sigma V(\tau + \ln|\mathcal{C}|)}{n^\alpha r^{2-\vartheta}}} + \frac{2c_B B(\tau + \ln|\mathcal{C}|)}{n^\alpha r}\right) + 2\varepsilon$$

with probability $\mu$ not less than $1 - C\,e^{-\tau}$. By combining this estimate with (12) and (16), we then obtain that

$$\Upsilon(f_{D_n,\Upsilon}) + \mathbb{E}_P h_{\bar{f}_{D_n},\Upsilon} < \Upsilon(f_0) + 2\mathbb{E}_P h_{f_0} + \left(\frac{c_\sigma V \tau}{n^\alpha}\right)^{\frac{1}{2-\vartheta}} + \frac{2c_B B \tau}{n^\alpha} + \frac{7c_B B_0 \tau}{4n^\alpha} + 2\varepsilon + \delta$$

$$+ \left(\mathbb{E}_P h_{\bar{f}_{D_n},\Upsilon} + \varepsilon + r\right)\left(\sqrt{\frac{c_\sigma V(\tau + \ln|\mathcal{C}|)}{n^\alpha r^{2-\vartheta}}} + \frac{2c_B B(\tau + \ln|\mathcal{C}|)}{n^\alpha r}\right) \tag{19}$$

holds with probability $\mu$ not less than $1 - 3Ce^{-\tau}$. Consequently, it remains to bound the various terms. To this end, we first observe that for

$$r := \left(\frac{36 c_\sigma V(\tau + \ln|\mathcal{C}|)}{n^\alpha}\right)^{1/(2-\vartheta)},$$

we obtain, since $6 \leq 36^{1/(2-\vartheta)}$,

$$\left(\frac{c_\sigma V \tau}{n^\alpha}\right)^{\frac{1}{2-\vartheta}} \leq \frac{r}{6} \qquad \text{and} \qquad \sqrt{\frac{c_\sigma V(\tau + \ln|\mathcal{C}|)}{n^\alpha r^{2-\vartheta}}} \leq \frac{1}{6}.$$

In addition, $V \geq B^{2-\vartheta}$, $c_\sigma \geq 3c_B$, $6 \leq 36^{1/(2-\vartheta)}$, and $n^\alpha \geq 3c_B(\tau + \ln|\mathcal{C}|)$ imply

$$\frac{2c_B B(\tau + \ln|\mathcal{C}|)}{rn^\alpha} = \frac{6}{9} \cdot \frac{3c_B(\tau + \ln|\mathcal{C}|)}{n^\alpha} \cdot \frac{B}{r} \leq \frac{6}{9} \cdot \left(\frac{3c_B(\tau + \ln|\mathcal{C}|)}{n^\alpha}\right)^{\frac{1}{2-\vartheta}} \cdot \frac{V^{\frac{1}{2-\vartheta}}}{r}$$

$$\leq \frac{1}{9} \cdot \left(\frac{36 c_\sigma V(\tau + \ln|\mathcal{C}|)}{n^\alpha r^{2-\vartheta}}\right)^{\frac{1}{2-\vartheta}} = \frac{1}{9}.$$

Using these estimates together with $1/6 + 1/9 \leq 1/3$ in (19), we see that

$$\Upsilon(f_{D_n,\Upsilon}) + \mathbb{E}_P h_{\bar{f}_{D_n},\Upsilon} < \Upsilon(f_0) + 2\mathbb{E}_P h_{f_0} + \frac{r}{3} + \frac{7c_B B_0 \tau}{4n^\alpha} + \frac{\mathbb{E}_P h_{\bar{f}_{D_n},\Upsilon} + \varepsilon + r}{3} + 2\varepsilon + \delta$$

holds with probability $\mu$ not less than $1 - 3Ce^{-\tau}$. Consequently, we have

$$\Upsilon(f_{D_n,\Upsilon}) + \mathbb{E}_P h_{\bar{f}_{D_n},\Upsilon} < 3\Upsilon(f_0) + 3\mathbb{E}_P h_{f_0} + \left(\frac{36 c_\sigma V(\tau + \ln|\mathcal{C}|)}{n^\alpha}\right)^{1/(2-\vartheta)} + \frac{4c_B B_0 \tau}{n^\alpha} + 4\varepsilon + 2\delta,$$

i.e. we have shown the assertion. ∎

***Proof of Corollary 2.3:*** The result follows from minimizing the right-hand side of the oracle inequality of Theorem 2.1 with respect to $\varepsilon$. ∎

## Footnotes

[1]For example, [4] write on page 71: "... it is a common practice to assume a certain mild asymptotic independence (such as $\alpha$-mixing) as a precondition in the context of ... nonlinear times series."

## References

[1] P. L. Bartlett, O. Bousquet, and S. Mendelson. Local Rademacher complexities. *Ann. Statist.*, 33:1497–1537, 2005.

[2] G. Blanchard, G. Lugosi, and N. Vayatis. On the rate of convergence of regularized boosting classifiers. *J. Mach. Learn. Res.*, 4:861–894, 2003.

[3] R. C. Bradley. *Introduction to Strong Mixing Conditions. Vol. 1-3.* Kendrick Press, Heber City, UT, 2007.

[4] J. Fan and Q. Yao. *Nonlinear Time Series.* Springer, New York, 2003.

[5] A. Irle. On consistency in nonparametric estimation under mixing conditions. *J. Multivariate Anal.*, 60:123–147, 1997.

[6] W. S. Lee, P. L. Bartlett, and R. C. Williamson. The importance of convexity in learning with squared loss. *IEEE Trans. Inform. Theory*, 44:1974–1980, 1998.

[7] A. Lozano, S. Kulkarni, and R. Schapire. Convergence and consistency of regularized boosting algorithms with stationary $\beta$-mixing observations. In Y. Weiss, B. Schölkopf, and J. Platt, editors, *Advances in Neural Information Processing Systems 18*, pages 819–826. MIT Press, Cambridge, MA, 2006.

[8] R. Meir. Nonparametric time series prediction through adaptive model selection. *Mach. Learn.*, 39:5–34, 2000.

[9] D. S. Modha and E. Masry. Minimum complexity regression estimation with weakly dependent observations. *IEEE Trans. Inform. Theory*, 42:2133–2145, 1996.

[10] M. Mohri and A. Rostamizadeh. Stability bounds for non-i.i.d. processes. In J.C. Platt, D. Koller, Y. Singer, and S. Roweis, editors, *Advances in Neural Information Processing Systems 20*, pages 1025–1032. MIT Press, Cambridge, MA, 2008.

[11] M. Mohri and A. Rostamizadeh. Rademacher complexity bounds for non-i.i.d. processes. In D. Koller, D. Schuurmans, Y. Bengio, and L. Bottou, editors, *Advances in Neural Information Processing Systems 21*, pages 1097–1104. 2009.

[12] I. Steinwart. Two oracle inequalities for regularized boosting classiers. *Statistics and Its Interface*, 2:271284, 2009.

[13] I. Steinwart and A. Christmann. *Support Vector Machines.* Springer, New York, 2008.

[14] I. Steinwart and A. Christmann. Estimating conditional quantiles with the help of the pinball loss. *Bernoulli*, accepted with minor revision.

[15] I. Steinwart, D. Hush, and C. Scovel. Learning from dependent observations. *J. Multivariate Anal.*, 100:175–194, 2009.

[16] I. Steinwart, D. Hush, and C. Scovel. Optimal rates for regularized least squares regression. In S. Dasgupta and A. Klivans, editors, *Proceedings of the 22nd Annual Conference on Learning Theory*, pages 79–93. 2009.

[17] H. Sun and Q. Wu. Regularized least square regression with dependent samples. *Adv. Comput. Math.*, to appear.

[18] M. Vidyasagar. *A Theory of Learning and Generalization: With Applications to Neural Networks and Control Systems.* Springer, London, 2nd edition, 2003.

[19] Y.-L. Xu and D.-R. Chen. Learning rates of regularized regression for exponentily strongly mixing sequence. *J. Statist. Plann. Inference*, 138:2180–2189, 2008.

[20] B. Yu. Rates of convergence for empirical processes of stationary mixing sequences. *Ann. Probab.*, 22:94–116, 1994.

[21] B. Zou and L. Li. The performance bounds of learning machines based on exponentially strongly mixing sequences. *Comput. Math. Appl.*, 53:1050–1058, 2007.

